# Text Classification using String Kernels

Huma Lodhi          John Shawe-Taylor          Nello Cristianini

**Chris Watkins**
Department of Computer Science Royal Holloway, University of London
Egham, Surrey TW20 0EX, UK
{huma, john, nello, chrisw}@dcs.rhbnc.ac.uk

## Abstract

We introduce a novel kernel for comparing two text documents. The kernel is an inner product in the feature space consisting of all subsequences of length $k$. A subsequence is any ordered sequence of $k$ characters occurring in the text though not necessarily contiguously. The subsequences are weighted by an exponentially decaying factor of their full length in the text, hence emphasising those occurrences which are close to contiguous. A direct computation of this feature vector would involve a prohibitive amount of computation even for modest values of $k$, since the dimension of the feature space grows exponentially with $k$. The paper describes how despite this fact the inner product can be efficiently evaluated by a dynamic programming technique. A preliminary experimental comparison of the performance of the kernel compared with a standard word feature space kernel [6] is made showing encouraging results.

## 1   Introduction

Standard learning systems (like neural networks or decision trees) operate on input data after they have been transformed into feature vectors $x_1, ..., x_\ell \in X$ from an $n$ dimensional space. There are cases, however, where the input data can not be readily described by explicit feature vectors: for example biosequences, images, graphs and text documents. For such datasets, the construction of a feature extraction module can be as complex and expensive as solving the entire problem. An effective alternative to explicit feature extraction is provided by kernel methods.

Kernel-based learning methods use an implicit mapping of the input data into a high dimensional feature space defined by a kernel function, i.e. a function returning the inner product between the images of two data points in the feature space. The learning then takes place in the feature space, provided the learning algorithm can be entirely rewritten so that the data points only appear inside dot products with other data points.

Several linear algorithms can be formulated in this way, for clustering, classification and regression. The most typical example of kernel-based systems is the Support

Vector Machine (SVM) [10, 3], that implements linear classification.

One interesting property of kernel-based systems is that, once a valid kernel function has been selected, one can practically work in spaces of any dimensionality without paying any computational cost, since the feature mapping is never effectively performed. In fact, one does not even need to know what features are being used. In this paper we examine the use of a kernel method based on string alignment for text categorization problems.

A standard approach [5] to text categorisation makes use of the so-called bag of words (BOW) representation, mapping a document to a bag (i.e. a set that counts repeated elements), hence losing all the word order information and only retaining the frequency of the terms in the document. This is usually accompanied by the removal of non-informative words (stop words) and by the replacing of words by their stems, so losing inflection information. This simple technique has recently been used very successfully in supervised learning tasks with Support Vector Machines (SVM) [5].

In this paper we propose a radically different approach, that considers documents simply as symbol sequences, and makes use of specific kernels. The approach is entirely subsymbolic, in the sense that it considers the document just like a unique long sequence, and still it is capable to capture topic information. We build on recent advances [11, 4] that demonstrated how to build kernels over general structures like sequences. The most remarkable property of such methods is that they map documents to vectors without explicitly representing them, by means of sequence alignment techniques. A dynamic programming technique makes the computation of the kernels very efficient (linear in the documents length).

It is surprising that such a radical strategy, only extracting allignment information, delivers positive results in topic classification, comparable with the performance of problem-specific strategies: it seems that in some sense the semantic of the document can be at least partly captured by the presence of certain substrings of symbols.

Support Vector Machines [3] are linear classifiers in a kernel defined feature space. The kernel is a function which returns the dot product of the feature vectors $\phi(x)$ and $\phi(x')$ of two inputs $x$ and $x'$ $K(x, x') = \phi(x)^T \phi(x')$. Choosing very high dimensional feature spaces ensures that the required functionality can be obtained using linear classifiers. The computational difficulties of working in such feature spaces is avoided by using a dual representation of the linear functions in terms of the training set $S = \{(x_1, y_1), (x_2, y_2), \ldots, (x_m, y_m)\}$,

$$f(x) = \sum_{i=1}^{m} \alpha_i y_i K(x, x_i) - b.$$

The danger of overfitting by resorting to such a high dimensional space is averted by maximising the margin or a related soft version of this criterion, a strategy that has been shown to ensure good generalisation despite the high dimensionality [9, 8].

## 2  A Kernel for Text Sequences

In this section we describe a kernel between two text documents. The idea is to compare them by means of the substrings they contain: the more substrings in common, the more similar they are. An important part is that such substrings do not need to be contiguous, and the degree of contiguity of one such substring in a document determines how much weight it will have in the comparison.

For example: the substring 'c-a-r' is present both in the word 'card' and in the word 'custard', but with different weighting. For each such substring there is a dimension of the feature space, and the value of such coordinate depends on how frequently and how compactly such string is embedded in the text. In order to deal with non-contiguous substrings, it is necessary to introduce a decay factor $\lambda \in (0,1)$ that can be used to weight the presence of a certain feature in a text (see Definition 1 for more details).

**Example.** Consider the words *cat, car, bat, bar*. If we consider only $k = 2$, we obtain an 8-dimensional feature space, where the words are mapped as follows:

|              | c-a         | c-t         | a-t         | b-a         | b-t         | c-r         | a-r         | b-r         |
|--------------|-------------|-------------|-------------|-------------|-------------|-------------|-------------|-------------|
| $\phi(\text{cat})$ | $\lambda^2$ | $\lambda^3$ | $\lambda^2$ | 0           | 0           | 0           | 0           | 0           |
| $\phi(\text{car})$ | $\lambda^2$ | 0           | 0           | 0           | 0           | $\lambda^3$ | $\lambda^2$ | 0           |
| $\phi(\text{bat})$ | 0           | 0           | $\lambda^2$ | $\lambda^2$ | $\lambda^3$ | 0           | 0           | 0           |
| $\phi(\text{bar})$ | 0           | 0           | 0           | $\lambda^2$ | 0           | 0           | $\lambda^2$ | $\lambda^3$ |

Hence, the unnormalized kernel between *car* and *cat* is $K(\text{car,cat}) = \lambda^4$, wherease the normalized version is obtained as follows: $K(\text{car,car}) = K(\text{cat,cat}) = 2\lambda^4 + \lambda^6$ and hence $K'(\text{car,cat}) = \lambda^4/(2\lambda^4 + \lambda^6) = 1/(2 + \lambda^2)$. Note that in general the document will contain more than one word, but the mapping for the whole document is into one feature space. Punctuation is ignored, but spaces are retained.

However, for interesting substring sizes (eg $> 4$) direct computation of all the relevant features would be impractical even for moderately sized texts and hence explicit use of such representation would be impossible. But it turns out that a kernel using such features can be defined and calculated in a very efficient way by using dynamic progamming techniques.

We derive the kernel by starting from the features and working out their inner product. In this case there is no need to prove that it satisfies Mercer's conditions (symmetry and positive semi-definiteness) since they will follow automatically from its definition as an inner product. This kernel is based on work [11, 4] mostly motivated by bioinformatics applications. It maps strings to a feature vector indexed by all $k$ tuples of characters. A $k$-tuple will have a non-zero entry if it occurs as a subsequence anywhere (not necessarily contiguously) in the string. The weighting of the feature will be the sum over the occurrences of the $k$-tuple of a decaying factor of the length of the occurrence.

**Definition 1** (String subsequence kernel) *Let $\Sigma$ be a finite alphabet. A string is a finite sequence of characters from $\Sigma$, including the empty sequence. For strings $s, t$, we denote by $|s|$ the length of the string $s = s_1 \ldots s_{|s|}$, and by $st$ the string obtained by concatenating the strings $s$ and $t$. The string $s[i : j]$ is the substring $s_i \ldots s_j$ of $s$. We say that $u$ is a subsequence of $s$, if there exist indices $\mathbf{i} = (i_1, \ldots, i_{|u|})$, with $1 \leq i_1 < \cdots < i_{|u|} \leq |s|$, such that $u_j = s_{i_j}$, for $j = 1, \ldots, |u|$, or $u = s[\mathbf{i}]$ for short. The length $l(\mathbf{i})$ of the subsequence in $s$ is $i_{|u|} - i_1 + 1$. We denote by $\Sigma^n$ the set of all finite strings of length $n$, and by $\Sigma^*$ the set of all strings*

$$\Sigma^* = \bigcup_{n=0}^{\infty} \Sigma^n. \tag{1}$$

*We now define feature spaces $F_n = \mathbb{R}^{\Sigma^n}$. The feature mapping $\phi$ for a string $s$ is given by defining the $u$ coordinate $\phi_u(s)$ for each $u \in \Sigma^n$. We define*

$$\phi_u(s) = \sum_{\mathbf{i}:u=s[\mathbf{i}]} \lambda^{l(\mathbf{i})}, \tag{2}$$

*for some $\lambda \leq 1$. These features measure the number of occurrences of subsequences in the string $s$ weighting them according to their lengths. Hence, the inner product of the feature vectors for two strings $s$ and $t$ give a sum over all common subsequences weighted according to their frequency of occurrence and lengths*

$$
\begin{aligned}
K_n(s,t) &= \sum_{u \in \Sigma^n} \langle \phi_u(s) \cdot \phi_u(t) \rangle = \sum_{u \in \Sigma^n} \sum_{\mathbf{i}:u=s[\mathbf{i}]} \lambda^{l(\mathbf{i})} \sum_{\mathbf{j}:u=t[\mathbf{j}]} \lambda^{l(\mathbf{j})} \\
&= \sum_{u \in \Sigma^n} \sum_{\mathbf{i}:u=s[\mathbf{i}]} \sum_{\mathbf{j}:u=t[\mathbf{j}]} \lambda^{l(\mathbf{i})+l(\mathbf{j})}.
\end{aligned}
$$

In order to derive an effective procedure for computing such kernel, we introduce an additional function which will aid in defining a recursive computation for this kernel. Let

$$
\begin{aligned}
K_i'(s,t) &= \sum_{u \in \Sigma^i} \sum_{\mathbf{i}:u=s[\mathbf{i}]} \sum_{\mathbf{j}:u=t[\mathbf{j}]} \lambda^{|s|+|t|-i_1-j_1+2}, \\
i &= 1,\dots,n-1,
\end{aligned}
$$

that is counting the length to the end of the strings $s$ and $t$ instead of just $l(\mathbf{i})$ and $l(\mathbf{j})$. We can now define a recursive computation for $K_i'$ and hence compute $K_n$,

**Definition 2** *Recursive computation of the subsequence kernel.*

$$
\begin{aligned}
K_0'(s,t) &= 1, \text{ for all } s,t, \\
K_i'(s,t) &= 0, \text{ if } \min(|s|,|t|) < i, \\
K_i(s,t) &= 0, \text{ if } \min(|s|,|t|) < i, \\
K_i'(sx,t) &= \lambda K_i'(s,t) + \sum_{j:t_j=x} K_{i-1}'(s,t[1:j-1])\lambda^{|t|-j+2}, \\
& \quad i = 1,\dots,n-1, \\
K_n(sx,t) &= K_n(s,t) + \sum_{j:t_j=x} K_{n-1}'(s,t[1:j-1])\lambda^2.
\end{aligned}
$$

The correctness of this recursion follows from observing how the length of the strings has increased, incurring a factor of $\lambda$ for each extra character, until the full length of $n$ characters has been attained. If we wished to compute $K_n(s,t)$ for a range of values of $n$, we would simply perform the computation of $K_i'(s,t)$ up to one less than the largest $n$ required, and then apply the last recursion for each $K_n(s,t)$ that is needed using the stored values of $K_i'(s,t)$. We can of course create a kernel $K(s,t)$ that combines the different $K_n(s,t)$ giving different (positive) weightings for each $n$. Once we have create such a kernel it is natural to normalise to remove any bias introduced by document length. We can produce this effect by normalising the feature vectors in the feature space. Hence, we create a new embedding $\hat{\phi}(s) = \frac{\phi(s)}{\|\phi(s)\|}$, which gives rise to the kernel

$$
\begin{aligned}
\hat{K}(s,t) &= \langle \hat{\phi}(s) \cdot \hat{\phi}(t) \rangle = \left\langle \frac{\phi(s)}{\|\phi(s)\|} \cdot \frac{\phi(t)}{\|\phi(t)\|} \right\rangle \\
&= \frac{1}{\|\phi(s)\| \|\phi(t)\|} \langle \phi(s) \cdot \phi(t) \rangle = \frac{K(s,t)}{\sqrt{K(s,s)K(t,t)}}
\end{aligned}
$$

The normalised kernel introduced above was implemented using the recursive formulas described above. The next section gives some more details of the algorithmics and this is followed by a section describing the results of applying the kernel in a Support Vector Machine for text classification.

## 3 Algorithmics

In this section we describe how special design techniques provide a significant speed-up of the procedure, by both accelerating the kernel evaluations and reducing their number.

We used a simple gradient based implementation of SVMs (see [3]) with a fixed threshold. In order to deal with large datasets, we used a form of chunking: beginning with a very small subset of the data and gradually building up the size of the training set, while ensuring that only points which failed to meet margin 1 on the current hypothesis were included in the next chunk.

Since each evaluation of the kernel function requires not neglectable computational resources, we designed the system so to only calculate those entries of the kernel matrix that are actually required by the training algorithm. This can significantly reduce the training time, since only a relatively small part of the kernel matrix is actually used by our implementation of SVM.

Special care in the implementation of the kernel described in Definition 1 can significantly speed-up its evaluation. As can be seen from the description of the recursion in Definition 2, its computation takes time proportional to $n|s||t|^2$, as the outermost recursion is over the sequence length and for each length and each additional character in $s$ and $t$ a sum over the sequence $t$ must be evaluated.

The complexity of the computation can be reduced to $O(n|s||t|)$, by first evaluating

$$K_i''(sx, t) = \sum_{j:t_j=x} K_{i-1}'(s, t[1:j-1])\lambda^{|t|-j+2}$$

and observing that we can then evaluate $K_i'(s,t)$ with the $O(|s||t|)$ recursion,

$$K_i'(sx, t) = \lambda K_i'(s, t) + K_i''(sx, t).$$

Now observe that $K_i''(sx, tu) = \lambda^{|u|}K_i''(sx, t)$, provided $x$ does not occur in $u$, while

$$K_i''(sx, tx) = \lambda \left( K_i''(sx, t) + \lambda K_{i-1}'(s, t) \right).$$

These observations together give an $O(|s||t|)$ recursion for computing $K_i''(s,t)$. Hence, we can evaluate the overall kernel in $O(n|s||t|)$ time.

## 4 Experimental Results

Our aim was to test the efficacy of this new approach to feature extraction for text categorization, and to compare with a state–of-the-art system such as the one used in [6]. Expecially, we wanted to see how the performance is affected by the tunable parameter $k$ (we have used values 3, 5 and 6). As expected, using longer substrings in the comparison of two documents gives an improved performance.

We used the same dataset as that reported in [6], namely the Reuters-21578 [7], as well as the Medline doucment collection of 1033 document abstracts from the National Library of Medicine. We performed all of our experiments on a subset of four categories, 'earn', 'acq', 'crude', and 'corn'.

A confusion matrix can be used to summarize the performance of the classifier (number of true/false positives/negatives):

|   | P | N |
|---|----|----|
| P | TP | FP |
| N | FN | TN |

We define precision: $P = \frac{TP}{TP+FP}$ and recall:$R = \frac{TP}{TP+FN}$. We then define the quantitiy $F1 = \frac{2PR}{P+R}$ to measure the performance of the classifier.

We applied the two different kernels to a subset of Reuters of 380 training examples and 90 test examples. The only difference in the experiments was the kernel used. The splits of the data were had the following sizes and numbers of positive examples in training and test sets: numbers of positive examples in training (testing) set out of 370 (90): earn 152 (40); 114 (25); 76 (15); 38 (10) in the Reuters database.

The preliminary experiment used different values of $k$, in order to identify the optimal one, with the category 'earn'. The follwing experiments all used a sequence length of 5 for the string subsequences kernel. We set $\lambda = 0.5$. The results obtained are shown in the following where the precision, recall and F1 values are shown for both kernels.

|       | F1    | Precision | Recall | # SV |
|-------|-------|-----------|--------|------|
| 3 S-K | 0.925 | 0.981     | 0.878  | 138  |
| 5 S-K | 0.936 | 0.992     | 0.888  | 237  |
| 6 S-K | 0.936 | 0.992     | 0.888  | 268  |
| W-K   | 0.925 | 0.989     | 0.867  | 250  |

Table 1: F1, Precision, Recall and number of Support Vectors for top reuter category earn averaged over 10 splits (n S-K ≡ string kernel of length n, W-K ≡ word kernel

|       | 5 S-K kernel | | | | W-K kernel | | | |
|-------|------|--------|--------|------|-------|--------|--------|------|
|       | F1    | Precis. | Recall | # SV | F1    | Precis. | Recall | # SV |
| earn  | 0.936 | 0.992  | 0.888  | 237  | 0.925 | 0.989  | 0.867  | 250  |
| acq   | 0.867 | 0.914  | 0.828  | 269  | 0.802 | 0.843  | 0.7680 | 276  |
| crude | 0.936 | 0.979  | 0.90   | 262  | 0.904 | 0.91   | 0.907  | 262  |
| corn  | 0.779 | 0.886  | 0.7    | 231  | 0.762 | 0.833  | 0.71   | 264  |

Table 2: Precision, Recall and F1 numbers for 4 categories for the two kernels: word kernel (W-K) and subsequences kernel (5 S-K)

The results are better in one category, similar or slightly better for the other categories. They certainly indicate that the new kernel can outperform the more classical approach, but equally the performance is not reliably better. The last table shows the results obtained for two categories in medLine data, numbers 20 and 23.

| Query | Train/Test | 3 S-K(#SV) | 5 S-K(#SV) | 6 S-K(#SV) | W-K(#SV) |
|-------|------------|------------|------------|------------|----------|
| #20   | 24/15      | 0.20 (101) | 0.637 (295) | 0.75 (386) | 0.235 (598) |
| #23   | 22/15      | 0.534 (107) | 0.409 (302) | 0.75 (382) | 0.636 (618) |

Table 3: F1 and number of Support Vectors for top two Medline queries

# 5 Conclusions

The paper has presented a novel kernel for text analysis, and tested it on a categorization task, which relies on evaluating an inner product in a very high dimensional feature space. For a given sequence length $k$ ($k = 5$ was used in the experiments reported) the features are indexed by all strings of length $k$. Direct computation of

all the relevant features would be impractical even for moderately sized texts. The paper has presented a dynamic programming style computation for computing the kernel directly from the input sequences without explicitly calculating the feature vectors.

Further refinements of the algorithm have resulted in a practical alternative to the more standard word feature based kernel used in previous SVM applications to text classification [6]. We have presented an experimental comparison of the word feature kernel with our subsequences kernel on a benchmark dataset with encouraging results. The results reported here are very preliminary and many questions remain to be resolved. First more extensive experiments are required to gain a more reliable picture of the performance of the new kernel, including the effect of varying the subsequence length and the parameter $\lambda$. The evaluation of the new kernel is still relatively time consuming and more research is needed to investigate ways of expediting this phase of the computation.

# References

[1] M. Aizerman, E. Braverman, and L. Rozonoer. Theoretical foundations of the potential function method in pattern recognition learning. *Automation and Remote Control*, 25:821–837, 1964.

[2] B. E. Boser, I. M. Guyon, and V. N. Vapnik. A training algorithm for optimal margin classifiers. In D. Haussler, editor, *Proceedings of the 5th Annual ACM Workshop on Computational Learning Theory*, pages 144–152. ACM Press, 1992.

[3] N. Cristianini and J. Shawe-Taylor. *An Introduction to Support Vector Machines*. Cambridge University Press, 2000. **www.support-vector.net**.

[4] D. Haussler. Convolution kernels on discrete structures. Technical Report UCSC-CRL-99-10, University of California in Santa Cruz, Computer Science Department, July 1999.

[5] T. Joachims. Text categorization with support vector machines: Learning with many relevant features. Technical Report 23, LS VIII, University of Dortmund, 1997.

[6] T. Joachims. Text categorization with support vector machines. In *Proceedings of European Conference on Machine Learning (ECML)*, 1998.

[7] David Lewis. Reuters-21578 collection. Technical report, Available at: `http://www.research.att.com/~lewis/reuters21578.html`, 1987.

[8] J. Shawe-Taylor and N. Cristianini Margin Distribution and Soft Margin In *Advances in Large Margin Classifiers*, MIT Press 2000.

[9] J. Shawe-Taylor, P. Bartlett, R. Williamson and M. Anthony Structural Risk Minimization over Data-Dependent Hierarchies In *EEE Transactions on Information Theory* 1998

[10] V. Vapnik. *Statistical Learning Theory*. Wiley, 1998.

[11] C. Watkins. Dynamic alignment kernels. Technical Report CSD-TR-98-11, Royal Holloway, University of London, Computer Science department, January 1999.
